# The *Effective* Number of Parameters: An Analysis of Generalization and Regularization in Nonlinear Learning Systems

**John E. Moody**
Department of Computer Science, Yale University
P.O. Box 2158 Yale Station, New Haven, CT 06520-2158
Internet: moody@cs.yale.edu, Phone: (203)432-1200

## Abstract

We present an analysis of how the generalization performance (expected test set error) relates to the expected training set error for nonlinear learning systems, such as multilayer perceptrons and radial basis functions. The principal result is the following relationship (computed to second order) between the expected test set and training set errors:

$$\langle \mathcal{E}_{test}(\lambda) \rangle_{\xi\xi'} \approx \langle \mathcal{E}_{train}(\lambda) \rangle_{\xi} + 2\sigma_{eff}^2 \frac{p_{eff}(\lambda)}{n} \ . \tag{1}$$

Here, $n$ is the size of the training sample $\xi$, $\sigma_{eff}^2$ is the effective noise variance in the response variable(s), $\lambda$ is a regularization or weight decay parameter, and $p_{eff}(\lambda)$ is the *effective number of parameters* in the nonlinear model. The expectations $\langle \ \rangle$ of training set and test set errors are taken over possible training sets $\xi$ and training and test sets $\xi'$ respectively. The effective number of parameters $p_{eff}(\lambda)$ usually differs from the true number of model parameters $p$ for nonlinear or regularized models; this theoretical conclusion is supported by Monte Carlo experiments. In addition to the surprising result that $p_{eff}(\lambda) \neq p$, we propose an estimate of (1) called the *generalized prediction error* $(GPE)$ which generalizes well established estimates of prediction risk such as Akaike's $FPE$ and $AIC$, Mallows $C_P$, and Barron's $PSE$ to the nonlinear setting.[1]

# 1  Background and Motivation

Many of the nonlinear learning systems of current interest for adaptive control, adaptive signal processing, and time series prediction, are supervised learning systems of the regression type. Understanding the relationship between generalization performance and training error and being able to estimate the generalization performance of such systems is of crucial importance. We will take the *prediction risk* (expected test set error) as our measure of generalization performance.

# 2  Learning from Examples

Consider a set of $n$ real-valued input/output data pairs $\xi(n) = \{\xi^i = (x^i, y^i); i = 1, \dots, n\}$ drawn from a stationary density $\Xi(\xi)$. The observations can be viewed as being generated according to the *signal plus noise* model[2]

$$y^i = \mu(x^i) + \epsilon^i \tag{2}$$

where $y^i$ is the observed response (dependent variable), $x^i$ are the independent variables sampled with input probability density $\Omega(x)$, $\epsilon^i$ is independent, identically-distributed (iid) noise sampled with density $\Phi(\epsilon)$ having mean 0 and variance $\sigma^2$,[3] and $\mu(x)$ is the *conditional mean*, an unknown function. From the signal plus noise perspective, the density $\Xi(\xi) = \Xi(x, y)$ can be represented as the product of two components, the conditional density $\Psi(y|x)$ and the input density $\Omega(x)$:

$$\begin{aligned} \Xi(x, y) &= \Psi(y|x)\,\Omega(x) \\ &\equiv \Phi(y - \mu(x))\,\Omega(x) \ . \end{aligned} \tag{3}$$

The learning problem is then to find an estimate $\widehat{\mu}(x)$ of the conditional mean $\mu(x)$ on the basis of the training set $\xi(n)$.

In many real world problems, few *a priori* assumptions can be made about the functional form of $\mu(x)$. Since a parametric function class is usually not known, one must resort to a *nonparametric regression* approach, whereby one constructs an estimate $\widehat{\mu}(x) = f(x)$ for $\mu(x)$ from a large class of functions $\mathcal{F}$ known to have good approximation properties (for example, $\mathcal{F}$ could be all possible radial basis function networks and multilayer perceptrons). The class of approximation functions is usually the union of a countable set of subclasses (specific network architectures)[4] $\mathcal{A} \subset \mathcal{F}$ for which the elements of each subclass $f(w, x) \in \mathcal{A}$ are continuously parametrized by a set of $p = p(\mathcal{A})$ weights $w = \{w^\alpha; \alpha = 1, \dots, p\}$. The task of finding the estimate $f(x)$ thus consists of two problems: choosing the best architecture $\widehat{\mathcal{A}}$ and choosing the best set of weights $\widehat{w}$ given the architecture. Note that in

the nonparametric setting, there does not typically exist a function $f(w^*, x) \in \mathcal{F}$ with a finite number of parameters such that $f(w^*, x) = \mu(x)$ for arbitrary $\mu(x)$. For this reason, the estimators $\widehat{\mu}(x) = f(\widehat{w}, x)$ will be *biased* estimators of $\mu(x)$.[5]

The first problem (finding the architecture $\mathcal{A}$) requires a search over possible architectures (*e.g.* network sizes and topologies), usually starting with small architectures and then considering larger ones. By necessity, the search is not usually exhaustive and must use heuristics to reduce search complexity. (A heuristic search procedure for two layer networks is presented in Moody and Utans (1992).)

The second problem (finding a good set of weights for $f(w, x)$) is accomplished by minimizing an objective function:

$$\widehat{w}_\lambda = \text{argmin}_w \, U(\lambda, w, \xi(n)) \ . \tag{4}$$

The objective function $U$ consists of an error function plus a regularizer:

$$U(\lambda, w, \xi(n)) = n \, \mathcal{E}_{train}(w, \xi(n)) + \lambda \, S(w) \tag{5}$$

Here, the error $\mathcal{E}_{train}(w, \xi(n))$ measures the "distance" between the target response values $y^i$ and the fitted values $f(w, x^i)$:

$$\mathcal{E}_{train}(w, \xi(n)) = \frac{1}{n} \sum_{i=1}^{n} \mathcal{E}[y^i, f(w, x^i)] \ , \tag{6}$$

and $S(w)$ is a regularization or weight-decay function which biases the solution toward functions with *a priori* "desirable" characteristics, such as smoothness. The parameter $\lambda \geq 0$ is the regularization or weight decay parameter and must itself be optimized.[6]

The most familiar example of an objective function uses the squared error[7] $\mathcal{E}[y^i, f(w, x^i)] = [y^i - f(w, x^i)]^2$ and a weight decay term:

$$U(\lambda, w, \xi(n)) = \sum_{i=1}^{n} (y^i - f(w, x^i))^2 + \lambda \sum_{\alpha=1}^{p} g(w^\alpha) \ . \tag{7}$$

The first term is the sum of squared errors ($SSE$) of the model $f(w, x)$ with respect to the training data, while the second term penalizes either small, medium, or large weights, depending on the form of $g(w^\alpha)$. Two common examples of weight decay functions are the ridge regression form $g(w^\alpha) = (w^\alpha)^2$ (which penalizes large weights) and the Rumelhart form $g(w^\alpha) = (w^\alpha)^2 / [(w^0)^2 + (w^\alpha)^2]$ (which penalizes weights of intermediate values near $w^0$).

An example of a regularizer which is not explicitly a weight decay term is:

$$S(w) = \int_x dx \Omega(x) \| \partial_{xx} f(w, x) \|^2 \ . \tag{8}$$

This is a smoothing term which penalizes functional fits with high curvature.

## 3   Prediction Risk

With $\hat{\mu}(x) = f(\hat{w}[\xi(n)], x)$ denoting an estimate of the true regression function $\mu(x)$ trained on a data set $\xi(n)$, we wish to estimate the prediction risk $P$, which is the expected error of $\hat{\mu}(x)$ in predicting future data. In principle, we can either define $P$ for models $\hat{\mu}(x)$ trained on arbitrary training sets of size $n$ sampled from the unknown density $\Psi(y|x)\Omega(x)$ or for training sets of size $n$ with input density equal to the empirical density defined by the single training set available:

$$\Omega'(x) = \frac{1}{n} \sum_{i=1}^{n} \delta(x - x^i) \ . \tag{9}$$

For such training sets, the $n$ inputs $x^i$ are held fixed, but the response variables $y^i$ are sampled with the conditional densities $\Psi(y|x^i)$. Since $\Omega'(x)$ is known, but $\Omega(x)$ is generally not known *a priori*, we adopt the latter approach.

For a large ensemble of such training sets, the *expected training set error* is[8]

$$
\begin{aligned}
\langle \mathcal{E}_{train}(\lambda) \rangle_\xi &= \left\langle \frac{1}{n} \sum_{i=1}^{n} \mathcal{E}[y^i, f(\hat{w}[\xi(n)], x^i)] \right\rangle_\xi \\
&= \int \frac{1}{n} \sum_{i=1}^{n} \mathcal{E}[y^i, f(\hat{w}[\xi(n)], x^i)] \left\{ \prod_{i=1}^{n} \Psi(y^i|x^i) dy^i \right\}
\end{aligned} \tag{10}
$$

For a future exemplar (x,z) sampled with density $\Psi(z|x)\Omega(x)$, the prediction risk $P$ is defined as:

$$P = \int \mathcal{E}[z, f(\hat{w}[\xi(n)], x)] \Psi(z|x)\Omega(x) \left\{ \prod_{i=1}^{n} \Psi(y^i|x^i) dy^i \right\} dz dx \tag{11}$$

Again, however, we don't assume that $\Omega(x)$ is known, so computing (11) is not possible.

Following Akaike (1970), Barron (1984), and numerous other authors (see Eubank 1988), we can define the prediction risk $P$ as the *expected test set error* for test sets of size $n$ $\xi'(n) = \{\xi^{i\prime} = (x^i, z^i); i = 1, \ldots, n\}$ having the empirical input density $\Omega'(x)$. This expected test set error has form:

$$
\begin{aligned}
\langle \mathcal{E}_{test}(\lambda) \rangle_{\xi\xi'} &= \left\langle \frac{1}{n} \sum_{i=1}^{n} \mathcal{E}[z^i, f(\hat{w}[\xi(n)], x^i)] \right\rangle_{\xi\xi'} \\
&= \int \frac{1}{n} \sum_{i=1}^{n} \mathcal{E}[z^i, f(\hat{w}[\xi(n)], x^i)] \left\{ \prod_{i=1}^{n} \Psi(y^i|x^i)\Psi(z^i|x^i) dy^i dz^i \right\}
\end{aligned} \tag{12}
$$

We take (12) as a proxy for the true prediction risk $P$.

In order to compute (12), it will not be necessary to know the precise functional form of the noise density $\Phi(\epsilon)$. Knowing just the noise variance $\sigma^2$ will enable an exact calculation for linear models trained with the $SSE$ error and an approximate calculation correct to second order for general nonlinear models. The results of these calculations are presented in the next two sections.

## 4    The Expected Test Set Error for Linear Models

The relationship between expected training set and expected test set errors for *linear models* trained using the $SSE$ error function with no regularizer is well known in statistics (Akaike 1970, Barron 1984, Eubank 1988). The exact relation for test and training sets with density (9):

$$\langle \mathcal{E}_{test} \rangle_{\xi\xi'} = \langle \mathcal{E}_{train} \rangle_{\xi} + 2\sigma^2 \frac{p}{n} \ . \tag{13}$$

As pointed out by Barron (1984), (13) can also apply approximately to the case of a nonlinear model $f(w, x)$ trained by minimizing the sum of squared errors $SSE$. This approximation can be arrived at in two ways. First, the model $f(\widehat{w}, x)$ can be treated as *locally linear* in a neighborhood of $\widehat{w}$. This approximation ignores the hessian and higher order shape of $f(w, x)$ in parameter space. Alternatively, the model $f(w, x)$ can be assumed to be *locally quadratic* in parameter space $w$ and *unbiased*.

However, the extension of (13) as an approximate relation for nonlinear models breaks down if any of the following situations hold:

The $SSE$ error function is not used. (For example, one may use a robust error measure (Huber 1981) or log likelihood error measure instead.)

A regularization term is included in the objective function. (This introduces bias.)

The *locally linear* approximation for $f(w, x)$ is not good.

The *unbiasedness* assumption for $f(w, x)$ is incorrect.

## 5    The Expected Test Set Error for Nonlinear Models

For neural network models, which are typically nonparametric (thus biased) and highly nonlinear, a new relationship is needed to replace (13). We have derived such a result correct to second order for nonlinear models:

$$\langle \mathcal{E}_{test}(\lambda) \rangle_{\xi\xi'} \approx \langle \mathcal{E}_{train}(\lambda) \rangle_{\xi} + 2\sigma^2_{eff} \frac{p_{eff}(\lambda)}{n} \ . \tag{14}$$

This result differs from the classical result (13) by the appearance of $p_{eff}(\lambda)$ (the *effective number of parameters*), $\sigma^2_{eff}$ (the effective noise variance in the response variable(s)), and a dependence on $\lambda$ (the regularization or weight decay parameter).

A full derivation of (14) will be presented in a longer paper (Moody 1992). The result is obtained by considering the noise terms $\epsilon^i$ for both the training and test

sets as perturbations to an idealized model fit to noise free data. The perturbative expansion is computed out to second order in the $\epsilon^i$'s subject to the constraint that the estimated weights $\hat{w}$ minimize the perturbed objective function. Computing expectation values and comparing the expansions for expected test and training errors yields (14). It is important to re-emphasize that deriving (14) does not require knowing the precise form of the noise density $\Phi(\epsilon)$. Only a knowledge of $\sigma^2$ is assumed.

The effective number of parameters $p_{eff}(\lambda)$ usually differs from the true number of model parameters $p$ and depends upon the amount of model bias, model non-linearity, and on our prior model preferences (eg. smoothness) as determined by the regularization parameter $\lambda$ and the form of our regularizer. The precise form of $p_{eff}(\lambda)$ is

$$p_{eff}(\lambda) \equiv \operatorname{tr} G \equiv \frac{1}{2} \sum_{i\alpha\beta} T_{i\alpha} U_{\alpha\beta}^{-1} T_{\beta i} \ , \tag{15}$$

where $G$ is the *generalized influence matrix* which generalizes the standard *influence* or *hat* matrix of linear regression, $T_{i\alpha}$ is the $n \times p$ matrix of derivatives of the training error function

$$T_{i\alpha} \equiv \frac{\partial}{\partial y^i} \frac{\partial}{\partial w^\alpha} n\mathcal{E}(w, \xi(n)) \ , \tag{16}$$

and $U_{\alpha\beta}^{-1}$ is the inverse of the hessian of the total objective function

$$U_{\alpha\beta} \equiv \frac{\partial}{\partial w^\alpha} \frac{\partial}{\partial w^\beta} \mathcal{U}(\lambda, w, \xi(n)) \ . \tag{17}$$

In the general case that $\sigma^2(x)$ varies with location in the input space $x$, the effective noise variance $\sigma_{eff}^2$ is a weighted average of the noise variances $\sigma^2(x^i)$. For the uniform signal plus noise model model we have described above, $\sigma_{eff}^2 = \sigma^2$.

## 6    The Effects of Regularization

In the neural network community, the most commonly used regularizers are weight decay functions. The use of weight decay is motivated by the intuitive notion that it removes unnecessary weights from the model. An analysis of $p_{eff}(\lambda)$ with weight decay ($\lambda > 0$) confirms this intuitive notion. Furthermore, whenever $\sigma^2 > 0$ and $n < \infty$, there exists some $\lambda_{optimal} > 0$ such that the expected test set error (12) is minimized. This is because weight decay methods yield models with lower model variance, even though they are biased. These effects will be discussed further in Moody (1992).

For models trained with squared error ($SSE$) and quadratic weight decay $g(w^\alpha) = (w^\alpha)^2$, the assumptions of unbiasedness[9] or local linearizability lead to the following expression for $p_{eff}(\lambda)$ which we call the *linearized effective* number of parameters $p_{lin}(\lambda)$:

$$p_{lin}(\lambda) = \sum_{\alpha=1}^{p} \frac{\kappa^\alpha}{\kappa^\alpha + \lambda} \ . \tag{18}$$

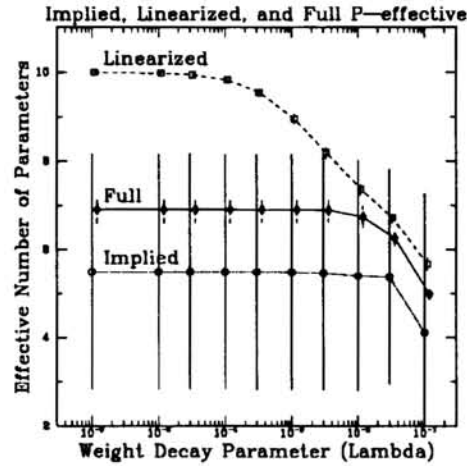

**Figure 1:** The full $p_{eff}(\lambda)$ (15) agrees with the implied $p_{imp}(\lambda)$ (19) to within experimental error, whereas the linearized $p_{lin}(\lambda)$ (18) does not (except for very large $\lambda$). These results verify the significance of (14) and (15) for nonlinear models.

Here, $\kappa^\alpha$ is the $\alpha^{th}$ eigenvalue of the $p \times p$ matrix $K = T^\dagger T$, with $T$ as defined in (16).

The form of $p_{eff}(\lambda)$ can be computed easily for other weight decay functions, such as the Rumelhart form $g(w^\alpha) = (w^\alpha)^2/[(w^0)^2 + (w^\alpha)^2]$. The basic result for all weight decay or regularization functions, however, is that $p_{eff}(\lambda)$ is a decreasing function of $\lambda$ with $p_{eff}(0) = p$ and $p_{eff}(\infty) = 0$, as is evident in the special case (18). If the model is nonlinear and biased, then $p_{eff}(0)$ generally differs from $p$.

## 7   Testing the Theory

To test the result (14) in a nonlinear context, we computed the full $p_{eff}(\lambda)$ (15), the linearized $p_{lin}(\lambda)$ (18), and the *implied number of parameters* $p_{imp}(\lambda)$ (19) for a nonlinear test problem. The value of $p_{imp}(\lambda)$ is obtained by computing the expected training and test errors for an ensemble of training sets of size $n$ with known noise variance $\sigma^2$ and solving for $p_{eff}(\lambda)$ in equation (14):

$$\widehat{p}_{imp}(\lambda) = \frac{n}{2\sigma^2}\left[\langle\widehat{\mathcal{E}_{test}(\lambda)}\rangle_{\xi\xi'} - \langle\widehat{\mathcal{E}_{train}(\lambda)}\rangle_\xi\right] \qquad (19)$$

The ^s indicate Monte Carlo estimates based on computations using a finite ensemble (10 in our experiments) of training sets. The test problem was to fit training sets of size 50 generated as a sum of three sigmoids plus noise, with the noise sampled from the uniform density. The model architecture $f(w, x)$ was also a sum of three sigmoids and the weights $\widehat{w}$ were estimated by minimizing (7) with quadratic weight decay. See figure 1.

## 8    GPE: An Estimate of Prediction Risk for Nonlinear Systems

A number of well established, closely related criteria for estimating the prediction risk for linear or linearizable models are available. These include Akaike's $FPE$ (1970), Akaike's $AIC$ (1973) Mallow's $C_P$ (1973), and Barron's $PSE$ (1984). (See also Akaike (1974) and Eubank (1988).) These estimates are all based on equation (13).

The generalized prediction error $(GPE)$ generalizes the classical estimators $FPE$, $AIC$, $C_P$, and $PSE$ to the nonlinear setting by estimating (14) as follows:

$$GPE(\lambda) = \widehat{P}_{GPE} = \mathcal{E}_{train}(n) + 2\widehat{\sigma}_{eff}^2 \frac{\widehat{p}_{eff}(\lambda)}{n} \ . \qquad (20)$$

The estimation process and the quality of the resulting $GPE$ estimates will be described in greater detail elsewhere.

### Acknowledgements

The author wishes to thank Andrew Barron and Joseph Chang for helpful conversations. This research was supported by AFOSR grant 89-0478 and ONR grant N00014-89-J-1228.

## Footnotes

[1] $GPE$ and $p_{eff}(\lambda)$ were previously introduced in Moody (1991).

[2]The assumption of additive noise $\epsilon$ which is independent of $x$ is a standard assumption and is not overly restrictive. Many other conceivable signal/noise models can be transformed into this form. For example, the multiplicative model $y = \mu(x)(1 + \epsilon)$ becomes $y' = \mu'(x) + \epsilon'$ for the transformed variable $y' = \log(y)$.

[3]Note that we have made only a minimal assumption about the noise $\epsilon$, that it is has finite variance $\sigma^2$ independent of $x$. Specifically, we do not need to make the assumption that the noise density $\Phi(\epsilon)$ is of known form (*e.g.* gaussian) for the following development.

[4]For example, a "fully connected two layer perceptron with five internal units".

[5]By *biased*, we mean that the mean squared bias is nonzero: $MSB = \int \rho(x)(\langle\widehat{\mu}(x)\rangle_\xi - \mu(x))^2 dx > 0$. Here, $\rho(x)$ is some positive weighting function on the input space and $\langle\rangle_\xi$ denotes an expected valued taken over possible training sets $\xi(n)$. For unbiasedness ($MSB = 0$) to occur, there must exist a set of weights $w^*$ such that $f(w^*, x) = \mu(x)$, and the learned weights $\widehat{w}$ must be "close to" $w^*$. For "near unbiasedness", we must have $w^* \equiv \text{argmin}_w MSB(w)$ such that $(MSB(w^*) \approx 0)$ and $\widehat{w}$ "close to" $w^*$.

[6]The optimization of $\lambda$ will be discussed in Moody (1992).

[7]Other error functions, such as those used in generalized linear models (see for example McCullagh and Nelder 1983) or robust statistics (see for example Huber 1981) are more appropriate than the squared error if the noise is known to be non-gaussian or the data contains many outliers.

[8]Following the physics convention, we use angled brackets $\langle \ \rangle$ to denote expected values. The subscripts denote the random variables being integrated over.

[9]Strictly speaking, a model with quadratic weight decay is unbiased only if the "true" weights are 0.

### References

H. Akaike. (1970). Statistical predictor identification. *Ann. Inst. Stat. Math.*, **22**:203.

H. Akaike. (1973). Information theory and an extension of the maximum likelihood principle. In *2nd Intl. Symp. on Information Theory*, Akademia Kiado, Budapest, 267.

H. Akaike. (1974). A new look at the statistical model identification. *IEEE Trans. Auto. Control*, **19**:716-723.

A. Barron. (1984). Predicted squared error: a criterion for automatic model selection. In *Self-Organizing Methods in Modeling*, S. Farlow, ed., Marcel Dekker, New York.

R. Eubank. (1988). *Spline Smoothing and Nonparametric Regression*. Marcel Dekker, New York.

P. J. Huber. (1981). *Robust Statistics*. Wiley, New York.

C. L. Mallows. (1973). Some comments on $C_P$. *Technometrics* **15**:661-675.

P. McCullagh and J.A. Nelder. (1983). *Generalized Linear Models*. Chapman and Hall, New York.

J. Moody. (1991). Note on Generalization, Regularization, and Architecture Selection in Nonlinear Learning Systems. In B.H. Juang, S.Y. Kung, and C.A. Kamm, editors, *Neural Networks for Signal Processing*, IEEE Press, Piscataway, NJ.

J. Moody. (1992). Long version of this paper, in preparation.

J. Moody and J. Utans. (1992). Principled architecture selection for neural networks: application to corporate bond rating prediction. In this volume.